# Generalized Prioritized Sweeping

**David Andre**   **Nir Friedman**   **Ronald Parr**
Computer Science Division, 387 Soda Hall
University of California, Berkeley, CA 94720
{dandre,nir,parr}@cs.berkeley.edu

## Abstract

*Prioritized sweeping* is a *model-based* reinforcement learning method that attempts to focus an agent's limited computational resources to achieve a good estimate of the value of environment states. To choose effectively where to spend a costly planning step, classic prioritized sweeping uses a simple heuristic to focus computation on the states that are likely to have the largest errors. In this paper, we introduce *generalized prioritized sweeping*, a principled method for generating such estimates in a representation-specific manner. This allows us to extend prioritized sweeping beyond an explicit, state-based representation to deal with compact representations that are necessary for dealing with large state spaces. We apply this method for generalized model approximators (such as Bayesian networks), and describe preliminary experiments that compare our approach with classical prioritized sweeping.

## 1  Introduction

In reinforcement learning, there is a tradeoff between spending time acting in the environment and spending time planning what actions are best. Model-free methods take one extreme on this question— the agent updates only the state most recently visited. On the other end of the spectrum lie classical dynamic programming methods that reevaluate the utility of every state in the environment after every experiment. Prioritized sweeping (PS) [6] provides a middle ground in that only the most "important" states are updated, according to a priority metric that attempts to measure the anticipated size of the update for each state. Roughly speaking, PS interleaves performing actions in the environment with propagating the values of states. After updating the value of state $s$, PS examines all states $t$ from which the agent might reach $s$ in one step and assigns them priority based on the expected size of the change in their value.

A crucial desideratum for reinforcement learning is the ability to scale-up to complex domains. For this, we need to use *compact* (or *generalizing*) representations of the model and the value function. While it is possible to apply PS in the presence of such representations (e.g., see [1]), we claim that classic PS is ill-suited in this case. With a generalizing model, a single experience may affect our estimation of the dynamics of many other states. Thus, we might want to update the value of states that are similar, in some appropriate sense, to $s$ since we have a new estimate of the system dynamics at these states. Note that some of these states might never have been reached before and standard PS will not assign them a priority at all.

In this paper, we present *generalized prioritized sweeping* (GenPS), a method that utilizes a formal principle to understand and extend PS and extend it to deal with parametric representations for both the model and the value function. If GenPS is used with an explicit state-space model and value function representation, an algorithm similar to the original (classic) PS results. When a model approximator (such as a *dynamic Bayesian network* [2]) is used, the resulting algorithm prioritizes the states of the environment using the generalizations inherent in the model representation.

## 2   The Basic Principle

We assume the reader is familiar with the basic concepts of *Markov Decision Processes* (MDPs); see, for example, [5]. We use the following notation: A MDP is a 4-tuple, $(S, A, p, r)$ where $S$ is a set of *states*, $A$ is a set of *actions*, $p(t \mid s, a)$ is a *transition model* that captures the probability of reaching state $t$ after we execute action $a$ at state $s$, and $r(s)$ is a *reward function* mapping $S$ into real-valued rewards. In this paper, we focus on infinite-horizon MDPs with a discount factor $\gamma$. The agent's aim is to maximize the expected discounted total reward it will receive. Reinforcement learning procedures attempt to achieve this objective when the agent *does not* know $p$ and $r$.

A standard problem in *model-based* reinforcement learning is one of balancing between planning (i.e., choosing a policy) and execution. Ideally, the agent would compute the optimal value function for its model of the environment each time the model changes. This scheme is unrealistic since finding the optimal policy for a given model is computationally non-trivial. Fortunately, we can approximate this scheme if we notice that the approximate model changes only slightly at each step. Thus, we can assume that the value function from the previous model can be easily "repaired" to reflect these changes. This approach was pursued in the DYNA [7] framework, where after the execution of an action, the agent updates its model of the environment, and then performs some bounded number of value propagation steps to update its approximation of the value function. Each value-propagation step locally enforces the *Bellman equation* by setting $\hat{V}(s) \leftarrow \max_{a \in A} \hat{Q}(s, a)$, where $\hat{Q}(s, a) = \hat{r}(s) + \gamma \sum_{s' \in S} \hat{p}(s' \mid s, a) \hat{V}(s')$, $\hat{p}(s' \mid s, a)$ and $\hat{r}(s)$ are the agent's approximation of the MDP, and $\hat{V}$ is the agent's approximation of the value function.

This raises the question of which states should be updated. In this paper we propose the following general principle:

> **GenPS Principle:** Update states where the approximation of the value function will change the most. That is, update the states with the largest *Bellman error*, $E(s) = |\hat{V}(s) - \max_{a \in A} \hat{Q}(s, a)|$.

The motivation for this principle is straightforward. The maximum Bellman error can be used to bound the maximum difference between the current value function, $\hat{V}(s)$ and the optimal value function, $V^*(s)$ [9]. This difference bounds the *policy loss*, the difference between the expected discounted reward received under the agent's current policy and the expected discounted reward received under the optimal policy.

To carry out this principle we have to recognize when the Bellman error at a state changes. This can happen at two different stages. First, after the agent updates its model of the world, new discrepancies between $\hat{V}(s)$ and $\max_a \hat{Q}(s, a)$ might be introduced, which can increase the Bellman error at $s$. Second, after the agent performs some value propagations, $\hat{V}$ is changed, which may introduce new discrepancies.

We assume that the agent maintains a value function and a model that are parameterized by $\theta_V$ and $\theta_M$. (We will sometimes refer to the vector that concatenates these vectors together into a single, larger vector simply as $\theta$.) When the agent observes a transition from state $s$ to $s'$ under action $a$, the agent updates its environment model by adjusting some of the parameters in $\theta_M$. When performing value-propagations, the agent updates $\hat{V}$ by updating parameters in $\theta_V$. A change in any of these parameters may change the Bellman error at other states in the model. We want to recognize these states without explicitly

computing the Bellman error at each one. Formally, we wish to estimate the change in error, $|\Delta_{E(s)}|$, due to the most recent change $\Delta_\theta$ in the parameters.

We propose approximating $|\Delta_{E(s)}|$ by using the gradient of the right hand side of the Bellman equation (i.e. $\max_a \hat{Q}(s,a)$). Thus, we have: $|\Delta_{E(s)}| \approx |\nabla \max_a \hat{Q}(s,a) \cdot \Delta_\theta|$ which estimates the change in the Bellman error at state $s$ as a function of the change in $\hat{Q}(s,a)$. The above still requires us to differentiate over a max, which is not differentiable. In general, we want to to overestimate the change, to avoid "starving" states with non-negligible error. Thus, we use the following upper bound: $|\nabla(\max_a \hat{Q}(s,a)) \cdot \Delta_\theta| \leq \max_a |\nabla \hat{Q}(s,a) \cdot \Delta_\theta|$.

We now define the generalized prioritized sweeping procedure. The procedure maintains a priority queue that assigns to each state $s$ a priority, $pri(s)$. After making some changes, we can reassign priorities by computing an approximation of the change in the value function.

Ideally, this is done using a procedure that implements the following steps:

>     procedure update-priorities ($\Delta_\theta$)
>         for all $s \in S$ $pri(s) \leftarrow pri(s) + \max_a |\nabla \hat{Q}(s,a) \cdot \Delta_\theta|$.

Note that when the above procedure updates the priority for a state that has an existing priority, the priorities are added together. This ensures that the priority being kept is an overestimate of the priority of each state, and thus, the procedure will eventually visit all states that require updating.

Also, in practice we would not want to reconsider the priority of all states after an update (we return to this issue below).

Using this procedure, we can now state the general learning procedure:

>     procedure GenPS ()
>         loop
>             perform an action in the environment
>             update the model; let $\Delta_\theta$ be the change in $\theta$
>             call update-priorities($\Delta_\theta$)
>             while there is available computation time
>                 let $s^{\max} = \arg\max_s pri(s)$
>                 perform value-propagation for $\hat{V}(s^{\max})$; let $\Delta_\theta$ be the change in $\theta$
>                 call update-priorities($\Delta_\theta$)
>                 $pri(s^{\max}) \leftarrow |\hat{V}(s^{\max}) - \max_a \hat{Q}(s^{\max},a)|$ [1]

Note that the GenPS procedure does not determine how actions are selected. This issue, which involves the problem of exploration, is orthogonal to the our main topic. Standard approaches, such as those described in [5, 6, 7], can be used with our procedure.

This abstract description specifies neither how to update the model, nor how to update the value function in the value-propagation steps. Both of these depend on the choices made in the corresponding representation of the model and the value function. Moreover, it is clear that in problems that involve a large state space, we cannot afford to recompute the priority of every state in update-priorities. However, we can simplify this computation by exploiting sparseness in the model and in the worst case we may resort to approximate methods for finding the states that receive high priority after each change.

## 3 Explicit, State-based Representation

In this section we briefly describe the instantiation of the generalized procedure when the rewards, values, and transition probabilities are explicitly modeled using lookup tables. In this representation, for each state $s$, we store the expected reward at $s$, denoted by $\theta_{\hat{r}(s)}$, the estimated value at $s$, denoted by $\theta_{\hat{V}(s)}$, and for each action $a$ and state $t$ the number of times the execution of $a$ at $s$ lead to state $t$, denoted $N_{s,a,t}$. From these transition counts we can

reconstruct the transition probabilities $\hat{p}(t \mid s,a) = \frac{N_{s,a,t}+N^0_{s,a,t}}{\sum_{t'} N_{s,a,t'}+N^0_{s,a,t'}}$, where $N^0_{s,a,t}$ are *fictional counts* that capture our prior information about the system's dynamics.[2] After each step in the world, these reward and probability parameters are updated in the straightforward manner. Value propagation steps in this representation set $\theta_{\hat{V}(t)}$ to the right hand side of the Bellman equation.

To apply the GenPS procedure we need to derive the gradient of the Bellman equation for two situations: (a) after a single step in the environment, and (b) after a value update.

In case (a), the model changes after performing action $s \xrightarrow{a} t$. In this case, it is easy to verify that $\nabla Q(s,a) \cdot \Delta_\theta = \Delta_{\theta_{\hat{r}(t)}} + \frac{\gamma}{\sum_t N_{s,a,t}+N^0_{s,a,t}} \left( V(t) - \sum_t' \hat{p}(t' \mid s,a)V(t') \right)$, and that $\nabla Q(s',a') \cdot \Delta_\theta = 0$ if $s' \neq s$ or $a' \neq a$. Thus, $s$ is the only state whose priority changes.

In case (b), the value function changes after updating the value of a state $t$. In this case, $\nabla Q(s,a) \cdot \Delta_\theta = \gamma \hat{p}(t \mid s,a)\Delta_{\theta_{\hat{V}(t)}}$. It is easy to see that this is nonzero only if $t$ is reachable from $s$. In both cases, it is straightforward to locate the states where the Bellman error might have have changed, and the computation of the new priority is more efficient than computing the Bellman-error.[3]

Now we can relate GenPS to standard prioritized sweeping. The PS procedure has the general form of this application of GenPS with three minor differences. First, after performing a transition $s \xrightarrow{a} t$ in the environment, PS immediately performs a value propagation for state $s$, while GenPS increments the priority of $s$. Second, after performing a value propagation for state $t$, PS updates the priority of states $s$ that can reach $t$ with the value $\max_a \hat{p}(t \mid s,a) \cdot \Delta_{\hat{V}(t)}$. The priority assigned by GenPS is the same quantity multiplied by $\gamma$. Since PS does not introduce priorities after model changes, this multiplicative constant does not change the order of states in the queue. Thirdly, GenPS uses addition to combine the old priority of a state with a new one, which ensures that the priority is indeed an upper bound. In contrast, PS uses max to combine priorities.

This discussion shows that PS can be thought of as a special case of GenPS when the agent uses an explicit, state-based representation. As we show in the next section, when the agent uses more compact representations, we get procedures where the prioritization strategy is quite different from that used in PS. Thus, we claim that classic PS is desirable primarily when explicit representations are used.

## 4  Factored Representation

We now examine a compact representation of $\hat{p}(s' \mid s,a)$ that is based on *dynamic Bayesian networks* (DBNs) [2]. DBNs have been combined with reinforcement learning before in [8], where they were used primarily as a means getting better generalization while learning. We will show that they also can be used with prioritized sweeping to focus the agent's attention on groups of states that are affected as the agent refines its environment model.

We start by assuming that the environment state is described by a set of *random variables*, $X_1, \ldots, X_n$. For now, we assume that each variable can take values from a finite set $Val(X_i)$. An *assignment* of values $x_1, \ldots, x_n$ to these variables describes a particular environment state. Similarly, we assume that the agent's action is described by random variables $A_1, \ldots, A_k$. To model the system dynamics, we have to represent the probability of transitions $s \xrightarrow{a} t$, where $s$ and $t$ are two assignments to $X_1, \ldots, X_n$ and $a$ is an assignment to $A_1, \ldots, A_k$. To simplify the discussion, we denote by $Y_1, \ldots, Y_n$ the agent's state after

the action is executed (e.g., the state $t$). Thus, $p(t \mid s, a)$ is represented as a conditional probability $P(Y_1, \ldots, Y_n \mid X_1, \ldots, X_n, A_1, \ldots, A_k)$.

A DBN model for such a conditional distribution consists of two components. The first is a directed acyclic graph where each vertex is labeled by a random variable and in which the vertices labeled $X_1, \ldots, X_n$ and $A_1, \ldots, A_k$ are roots. This graph specifies the *factorization* of the conditional distribution:

$$P(Y_1, \ldots, Y_n \mid X_1, \ldots, X_n, A_1, \ldots, A_k) = \prod_{i=1}^{n} P(Y_i \mid Pa_i), \tag{1}$$

where $Pa_i$ are the parents of $Y_i$ in the graph. The second component of the DBN model is a description of the conditional probabilities $P(Y_i \mid Pa_i)$. Together, these two components describe a unique conditional distribution. The simplest representation of $P(Y_i \mid Pa_i)$ is a table that contains a parameter $\theta_{i,y,z} = P(Y_i = y \mid Pa_i = z)$ for each possible combination of $y \in Val(Y_i)$ and $z \in Val(Pa_i)$ (note that $z$ is a joint assignment to several random variables). It is easy to see that the "density" of the DBN graph determines the number of parameters needed. In particular, a *complete* graph, to which we cannot add an arc without violating the constraints, is equivalent to a state-based representation in terms of the number of parameters needed. On the other hand, a sparse graph requires few parameters.

In this paper, we assume that the learner is supplied with the DBN structure and only has to learn the conditional probability entries. It is often easy to assess structure information from experts even when precise probabilities are not available. As in the state-based representation, we learn the parameters using Dirichlet priors for each multinomial distribution [4]. In this method, we assess the conditional probability $\theta_{i,y,z}$ using prior knowledge and the frequency of transitions observed in the past where $Y_i = y$ among those transitions where $Pa_i = z$. Learning amounts to keeping counts $N_{i,y,z}$ that record the number of transitions where $Y_i = y$ and $Pa_i = z$ for each variable $Y_i$ and values $y \in Val(Y_i)$ and $z \in Val(Pa_i)$. Our prior knowledge is represented by fictional counts $N^0_{i,y,z}$. Then we estimate probabilities using the formula $\theta_{i,y,z} = \frac{N_{i,y,z} + N^0_{i,y,z}}{N_{i,\cdot,z}}$, where $N_{i,\cdot,z} = \sum_{y'} N_{i,y',z} + N^0_{i,y',z}$.

We now identify which states should be reconsidered after we update the DBN parameters. Recall that this requires estimating the term $\nabla Q(s, a) \cdot \Delta_\theta$. Since $\Delta_\theta$ is sparse, after making the transition $s^* \xrightarrow{a^*} t^*$, we have that $\nabla Q(s, a) \cdot \Delta_\theta = \sum_i \frac{\partial Q(s,a)}{\partial N_{i,y_i^*,z_i^*}}$, where $y_i^*$ and $z_i^*$ are the assignments to $Y_i$ and $Pa_i$, respectively, in $s^* \xrightarrow{a^*} t^*$. (Recall that $s^*$, $a^*$ and $t^*$ jointly assign values to all the variables in the DBN.)

We say that a transition $s \xrightarrow{a} t$ is *consistent* with an assignment $X = x$ for a vector of random variables $X$, denoted $(s, a, t) \models (X = x)$, if $X$ is assigned the value $x$ in $s \xrightarrow{a} t$. We also need a similar notion for a partial description of a transition. We say that $s$ and $a$ are consistent with $X = x$, denoted $(s, a, \cdot) \models (X = x)$, if there is a $t$ such that $(s, a, t) \models (X = x)$.

Using this notation, we can show that if $(s, a, \cdot) \models (Pa_i = z_i^*)$, then

$$\frac{\partial Q(s,a)}{\partial N_{i,y_i^*,z_i^*}} = \frac{\gamma}{N_{i,\cdot,z_i^*}} \left[ \frac{1}{\theta_{i,y_i^*,z_i^*}} \sum_{t:(s,a,t)\models y_i^*, z_i^*} \hat{p}(t \mid s, a)\hat{V}(t) - \sum_{t:(s,a,t)\models z_i^*} \hat{p}(t \mid s, a)\hat{V}(t) \right],$$

and if $s, a$ are inconsistent with $Pa_i = z_i^*$, then $\frac{\partial Q(s,a)}{\partial N_{i,y_i^*,z_i^*}} = 0$.

This expression shows that if $s$ is similar to $s^*$ in that both agree on the values they assign to the parents of some $Y_i$ (i.e., $(s, a^*)$ is consistent with $z_i^*$), then the priority of $s$ would change after we update the model. The magnitude of the priority change depends upon both the similarity of $s$ and $s^*$ (i.e. how many of the terms in $\nabla Q(s, a) \cdot \Delta_\theta$ will be non-zero), and the value of the states that can be reached from $s$.

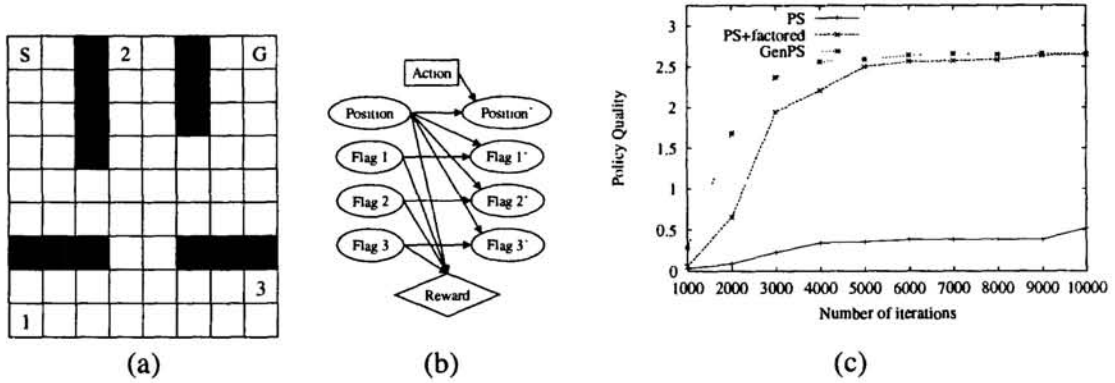

Figure 1: (a) The maze used in the experiment. S marks the start space, G the goal state, and 1, 2 and 3 are the three flags the agent has to set to receive the reward. (b) The DBN structure that captures the independencies in this domain. (c) A graph showing the performance of the three procedures on this example. PS is GenPS with a state-based model, PS+factored is the same procedure but with a factored model, and GenPS exploits the factored model in prioritization. Each curve is the average of 5 runs.

The evaluation of $\frac{\partial Q(s,a)}{\partial N_{i,y_i^*,z_i^*}}$ requires us to sum over a subset of the states –namely, those states $t$ that are consistent with $z_i^*$. Unfortunately, in the worst case this will be a large fragment of the state space. If the number of environment states is not large, then this might be a reasonable cost to pay for the additional benefits of GenPS. However, this might be a burdensome when we have a large state space, which are the cases where we expect to gain the most benefit from using generalized representations such as DBN.

In these situations, we propose a heuristic approach for estimating $\nabla Q(s,a)\Delta_\theta$ without summing over large numbers of states for computing the change of priority for each possible state. This can be done by finding upper bounds on or estimates of $\frac{\partial Q(s,a)}{\partial N_{i,y_i^*,z_i^*}}$. Once we have computed these estimates, we can estimate the priority change for each state $s$. We use the notation $s \sim_i s^*$ if $s$ and $s^*$ both agree on the assignment to $Pa_i$. If $C_i$ is an upper bound on (or an estimate of) $\left|\frac{\partial Q(s,a)}{\partial N_{i,y_i^*,z_i^*}}\right|$, we have that $|\nabla Q(s,a)\Delta_{\theta_M}| \leq \sum_{i:s\sim_i s^*} C_i$.

Thus, to evaluate the priority of state $s$, we simply find how "similar" it is to $s^*$. Note that it is relatively straightforward to use this equation to enumerate all the states where the priority change might be large. Finally, we note that the use of a DBN as a model does not change the way we update priorities after a value propagation step. If we use an explicit table of values, then we would update priorities as in the previous section. If we use a compact description of the value function, then we can apply GenPS to get the appropriate update rule.

# 5 An Experiment

We conducted an experiment to evaluate the effect of using GenPS with a generalizing model. We used a maze domain similar to the one described in [6]. The maze, shown in Figure 1(a), contains 59 cells, and 3 binary flags, resulting in $59 \times 2^3 = 472$ possible states. Initially the agent is at the start cell (marked by S) and the flags are reset. The agent has four possible actions, up, down, left, and right, that succeed 80% of the time, and 20% of the time the agent moves in an unintended perpendicular direction. The $i$'th flag is set when the agent leaves the cell marked by $i$. The agent receives a reward when it arrives at the goal cell (marked by G) and all of the flags are set. In this situation, any action resets the game. As noted in [6], this environment exhibits independencies. Namely, the probability of transition from one cell to another does not depend on the flag settings.

These independencies can be captured easily by the simple DBN shown in Figure 1(b) Our experiment is designed to test the extent to which GenPS exploits the knowledge of these independencies for faster learning.

We tested three procedures. The first is GenPS, which uses an explicit state-based model. As explained above, this variant is essentially PS. The second procedure uses a factored model of the environment for learning the model parameters, but uses the same prioritization strategy as the first one. The third procedure uses the GenPS prioritization strategy we describe in Section 4. All three procedures use the Boltzman exploration strategy (see for example [5]). Finally, in each iteration these procedures process at most 10 states from the priority queue.

The results are shown in Figure 1(c). As we can see, the GenPS procedure converged faster than the procedures that used classic PS. As we can see, by using the factored model we get two improvements. The first improvement is due to generalization in the model. This allows the agent to learn a good model of its environment after fewer iterations. This explains why PS+factored converges faster than PS. The second improvement is due to the better prioritization strategy. This explains the faster convergence of GenPS.

## 6 Discussion

We have presented a general method for approximating the optimal use of computational resources during reinforcement learning. Like classic prioritized sweeping, our method aims to perform only the most beneficial value propagations. By using the gradient of the Bellman equation our method generalizes the underlying principle in prioritized sweeping. The generalized procedure can then be applied not only in the explicit, state-based case, but in cases where approximators are used for the model. The generalized procedure also extends to cases where a function approximator (such as that discussed in [3]) is used for the value function, and future work will empirically test this application of GenPS. We are currently working on applying GenPS to other types of model and function approximators.

### Acknowledgments

We are grateful to Geoff Gordon, Daishi Harada, Kevin Murphy, and Stuart Russell for discussions related to this work and comments on earlier versions of this paper. This research was supported in part by ARO under the MURI program "Integrated Approach to Intelligent Systems," grant number DAAH04-96-1-0341. The first author is supported by a National Defense Science and Engineering Graduate Fellowship.

## Footnotes

[1] In general, this will assign the state a new priority of 0, unless there is a self loop. In this case it will easy to compute the new Bellman error as a by-product of the value propagation step.

[2]Formally, we are using multinomial *Dirichlet* priors. See, for example, [4] for an introduction to these Bayesian methods.

[3]Although $\frac{\partial \hat{Q}(s,a)}{\partial N_{s,a,t}}$ involves a summation over all states, it can be computed efficiently. To see this, note that the summation is essentially the old value of $Q(s,a)$ (minus the immediate reward) which can be retained in memory.

## References

[1] S. Davies. Multidimensional triangulation and interpolation for reinforcement learning. In *Advances in Neural Information Processing Systems 9.* 1996.

[2] T. Dean and K. Kanazawa. A model for reasoning about persistence and causation. *Computational Intelligence*, 5:142–150, 1989.

[3] G. J. Gordon. Stable function approximation in dynamic programming. In *Proc. 12th Int. Conf. on Machine Learning*, 1995.

[4] D. Heckerman. A tutorial on learning with Bayesian networks. Technical Report MSR-TR-95-06, Microsoft Research, 1995. Revised November 1996.

[5] L. P. Kaelbling, M. L. Littman and A. W. Moore. Reinforcement learning: A survey. *Journal of Artificial Intelligence Research*, 4:237–285, 1996.

[6] A. W. Moore and C. G. Atkeson. Prioritized sweeping—reinforcement learning with less data and less time. *Machine Learning*, 13:103–130, 1993.

[7] R. S. Sutton. Integrated architectures for learning, planning, and reacting based on approximating dynamic programming. In *Machine Learning: Proc. 7th Int. Conf.*, 1990.

[8] P. Tadepalli and D. Ok. Scaling up average reward reinforcement learning by approximating the domain models and the value function. In *Proc. 13th Int. Conf. on Machine Learning*, 1996.

[9] R. J. Williams and L. C. III Baird. Tight performance bounds on greedy policies based on imperfect value functions. Technical report, Computer Science, Northeastern University. 1993.